# Bayesian Backpropagation Over I-O Functions Rather Than Weights

**David H. Wolpert**
The Santa Fe Institute
1660 Old Pecos Trail
Santa Fe, NM 87501

## Abstract

The conventional Bayesian justification of backprop is that it finds the MAP weight vector. As this paper shows, to find the MAP i-o function instead one must add a correction term to backprop. That term biases one towards i-o functions with small description lengths, and in particular favors (some kinds of) feature-selection, pruning, and weight-sharing.

## 1 INTRODUCTION

In the conventional Bayesian view of backpropagation (BP) (Buntine and Weigend, 1991; Nowlan and Hinton, 1994; MacKay, 1992; Wolpert, 1993), one starts with the "likelihood" conditional distribution $P$(training set $= t$ | weight vector $w$) and the "prior" distribution $P(w)$. As an example, in regression one might have a "Gaussian likelihood", $P(t \mid w) \propto \exp[-\chi^2(w, t)] \equiv \Pi_i \exp [-\{net(w, t_X(i)) - t_y(i)\}^2 / 2\sigma^2]$ for some constant $\sigma$. ($t_X(i)$ and $t_Y(i)$ are the successive input and output values in the training set respectively, and $net(w, .)$ is the function, induced by $w$, taking input neuron values to output neuron values.) As another example, the "weight decay" (Gaussian) prior is $P(w) \propto \exp(-\alpha(w^2))$ for some constant $\alpha$.

Bayes' theorem tells us that $P(w \mid t) \propto P(t \mid w) P(w)$. Accordingly, the most probable weight given the data - the "maximum a posteriori" (MAP) $w$ - is the mode over $w$ of $P(t \mid w) P(w)$, which equals the mode over $w$ of the "cost function" $L(w, t) \equiv \ln[P(t \mid w)] + \ln[P(w)]$. So for example with the Gaussian likelihood and weight decay prior, the most probable $w$ given the data is the $w$ minimizing $\chi^2(w, t) + \alpha w^2$. Accordingly BP with weight decay can be viewed as a scheme for trying to find the function from input neuron values to output neuron values (i-o function) induced by the MAP $w$.

One peculiar aspect of this justification of weight-decay BP is the fact that rather than the i-o function induced by the most probable *weight vector*, in practice one would usually prefer to know the most probable i-o *function*. (In few situations would one care more about a weight vector than about what that weight vector parameterizes.) Unfortunately, the difference between these two i-o functions can be large; in general it is *not* true that "the most probable output corresponds to the most probable parameter" (Denker and LeCun, 1991).

This paper shows that to find the MAP i-o function rather than the MAP w one adds a "correction term" to conventional BP. That term biases one towards i-o functions with small description lengths, and in particular favors feature-selection, pruning and weight-sharing. In this that term constitutes a theoretical justification for those techniques.

Although cast in terms of neural nets, this paper's analysis applies to any case where convention is to use the MAP value of a parameter encoding Z to estimate the value of Z.

## 2     BACKPROP OVER I-O FUNCTIONS

Assume the net's architecture is fixed, and that weight vectors w live in a Euclidean vector space W of dimension $|W|$. Let X be the set of vectors x which can be loaded on the input neurons, and O the set of vectors o which can be read off the output neurons. Assume that the number of elements in X ($|X|$) is finite. This is always the case in the real world, where measuring devices have finite accuracy, and where the computers used to emulate neural nets are finite state machines. For similar reasons O is also finite in practice. However for now assume that O is very large and "fine-grained", and approximate it as a Euclidean vector space of dimension $|O|$. (This assumption usually holds with neural nets, where output values are treated as real-valued vectors.) This assumption will be relaxed later.

Indicate the set of functions taking X to O by $\Phi$. (net(w, .) is an element of $\Phi$.) Any $\phi \in \Phi$ is an ($|X| \times |O|$)-dimensional Euclidean vector. Accordingly, densities over W are related to densities over $\Phi$ by the usual rules for transforming densities between $|W|$-dimensional and ($|X| \times |O|$)-dimensional Euclidean vector spaces. There are three cases to consider:

1)       $|W| < |X| |O|$. In general, as one varies over all w's the corresponding i-o functions net(w, .) map out a sub-manifold of $\Phi$ having lower dimension than $\Phi$.
2)       $|W| > |X| |O|$. There are an infinite number of w's corresponding to each $\phi$.
3)       $|W| = |X| |O|$. This is the easiest case to analyze in detail. Accordingly I will deal with it first, deferring discussion of cases one and two until later.

With some abuse of notation, let capital letters indicate random variables and lower case letters indicate values of random variables. So for example w is a value of the weight vector random variable W. Use 'p' to indicate probability densities. So for example $p_{\Phi|T}(\phi \mid t)$ is the density of the i-o function random variable $\Phi$, conditioned on the training set random variable T, and evaluated at the values $\Phi = \phi$ and $T = t$.

In general, any i-o function not expressible as net(w, .) for some w has zero probability. For the other i-o functions, with $\delta(.)$ being the multivariable Dirac delta function,

$$p_{\Phi}(\text{net}(w, .)) = \int dw' \, p_W(w') \, \delta(\text{net}(w', .) - \text{net}(w, .)). \qquad (1)$$

When the mapping $\Phi = \text{net}(W, .)$ is one-to-one, we can evaluate equation (1) to get

$$p_{\Phi|T}(\text{net}(w, .) \mid t) = p_{W|T}(w \mid t) / J_{\Phi,W}(w), \qquad (2)$$

where $J_{\Phi,W}(w)$ is the Jacobian of the $W \rightarrow \Phi$ mapping:

$$J_{\Phi,W}(w) \equiv |\det[\, \partial\Phi_i / \partial W_j \,](w)| = |\det[\, \partial \, net(w, .)_i / \partial w_j \,]|. \qquad (3)$$

"$net(w, .)_i$" means the i'th component of the i-o function $net(w, .)$. "$net(w, x)$" means the vector o mapped by $net(w, .)$ from the input x, and "$net(w, x)_k$" is the k'th component of o. So the "i" in "$net(w, .)_i$" refers to a pair of values $\{x, k\}$. Each matrix value $\partial\phi_i / \partial w_j$ is the partial derivative of $net(w, x)_k$ with respect to some weight, for some x and k. $J_{\Phi,W}(w)$ can be rewritten as $\det^{1/2}[g_{ij}(w)]$, where $g_{ij}(w) \equiv \Sigma_k [(\partial\phi_k / \partial w_i) (\partial\phi_k / \partial w_j)]$ is the metric of the $W \to \Phi$ mapping. This form of $J_{\Phi,W}(w)$ is usually more difficult to evaluate though.

Unfortunately, $\phi = net(w, .)$ is not one-to-one; where $J_{\Phi,W}(w) \neq 0$ the mapping is *locally* one-to-one, but there are global symmetries which ensure that more than one w corresponds to each $\phi$. (Such symmetries arise from things like permuting the hidden neurons or changing the sign of all weights leading into a hidden neuron - see (Fefferman, 1993) and references therein.) To circumvent this difficulty we must make a pair of assumptions.

To begin, restrict attention to $W_{inj}$, those values w of the variable W for which the Jacobian is non-zero. This ensures local injectivity of the map between W and $\Phi$. Given a particular $w \in W_{inj}$, let k be the number of $w' \in W_{inj}$ such that $net(w, .) = net(w', .)$. (Since $net(w, .) = net(w, .)$, $k \geq 1$.) Such a set of k vectors form an equivalence class, $\{w\}$.

The first assumption is that for all $w \in W_{inj}$ the size of $\{w\}$ (i.e., k) is the same. This will be the case if we exclude degenerate w (e.g., w's with all first layer weights set to 0). The second assumption is that for all w' and w in the same equivalence class, $p_{W|D}(w \mid d) = p_{W|D}(w' \mid d)$. This is usually the case. (For example, start with w' and relabel hidden neurons to get a new $w \in \{w'\}$. If we assume the Gaussian likelihood and prior, then since neither differs for the two w's the weight-posterior is also the same for the two w's.) Given these assumptions, $p_{\Phi|T}(net(w, .) \mid t) = k \, p_{W|T}(w \mid t) / J_{\Phi,W}(w)$. So rather than minimize the usual cost function, $L(w, t)$, to find the MAP $\Phi$ BP should minimize $L'(w, t) \equiv L(w, t) + \ln[\, J_{\Phi,W}(w) \,]$. The $\ln[\, J_{\Phi,W}(w) \,]$ term constitutes a correction term to conventional BP.

One should not confuse the correction term with the other quantities in the neural net literature which involve partial derivative matrices. As an example, one way to characterize the "quality" of a local peak w' of a cost function involves the Hessian of that cost function (Buntine and Weigend, 1991). The correction term doesn't directly concern the validity of such a Hessian-based quality measure. However it does concern the validity of some implementations of such a measure. In particular, the correction term changes the location of the peak w'. It also suggests that a peak's quality be measured by the Hessian of $L'(w', t)$ with respect to $\phi$, rather than by the Hessian of $L(w', t)$ with respect to w. (As an aside on the subject of Hessians, note that some workers incorrectly use Hessians when they attempt to evaluate quantities like output-variances. See (Wolpert, 1994).)

If we stipulate that the $p_{\Phi|T}(\phi \mid t)$ one encounters in the real world is independent of how one chooses to parameterize $\Phi$, then the probability density of our parameter must depend on how it gets mapped to $\Phi$. This is the basis of the correction term. As this suggests, the correction term won't arise if we use non-$p_{\Phi|T}(\phi \mid t)$-based estimators, like maximum-likelihood estimators. (This is a basic difference between such estimators and MAP estimators with a uniform prior.) The correction term is also irrelevant if it we use an MAP estimate but $J_{\Phi,W}(w)$ is independent of w (as when $net(w, .)$ depends linearly on w). And for non-linear $net(w, .)$, the correction term has no effect for some non-MAP-based ways to apply Bayesianism to neural nets, like guessing the posterior average $\Phi$ (Neal, 1993):

$$E(\Phi \mid t) \equiv \int d\phi \; p_{\Phi|T}(\phi \mid t) \; \phi = \int dw \; p_{W|T}(w \mid t) \; net(w, .), \qquad (4)$$

so one can calculate $E(\Phi \mid t)$ by working in $W$, without any concern for a correction term. (Loosely speaking, the Jacobian associated with changing integration variables cancels the Jacobian associated with changing the argument of the probability density. A formal derivation - applicable even when $|W| \neq |X| \times |O|$ - is in the appendix of (Wolpert, 1994).)

One might think that since it's independent of t, the correction term can be absorbed into $p_W(w)$. Ironically, it is precisely because quantities like $E(\Phi \mid t)$ aren't affected by the correction term that this is impossible: Absorb the correction term into the prior, giving a new prior $p^*_W(w) \equiv d \times p_W(w) \times J_{\Phi,W}(w)$ (asterisks refers to new densities, and d is a normalization constant). Then $p^*_{\Phi|T}(net(w, .) \mid t) = p_{W|T}(w \mid t)$. So by redefining what we call the prior we can justify use of conventional uncorrected BP; the (new) MAP $\phi$ corresponds to the w minimizing $L(w, t)$. However such a redefinition changes $E(\Phi \mid t)$ (amongst other things): $\int d\phi \; p^*_{\Phi|T}(\phi \mid t) \; \phi = \int dw \; p^*_{W|T}(w \mid t) \; net(w, .) \neq \int dw \; p_{W|T}(w \mid t) \; net(w, .) = \int d\phi \; p_{\Phi|T}(\phi \mid t) \; \phi$. So one can either modify BP (by adding in the correction term) and leave $E(\Phi \mid t)$ alone, or leave BP alone but change $E(\Phi \mid t)$; one can not leave both unchanged.

Moreover, some procedures involve both prior-based modes and prior-based integrals, and therefore are affected by the correction term no matter how $p_W(w)$ is redefined. For example, in the evidence procedure (Wolpert, 1993; MacKay, 1992) one fixes the value of a hyperparameter $\Gamma$ (e.g., $\alpha$ from the introduction) to the value $\gamma$ maximizing $p_{\Gamma|T}(\gamma \mid t)$. Next one find the value s' maximizing $p_{S|T,\Gamma}(s' \mid t, \gamma)$ for some variable S. Finally, one guesses the $\phi$ associated with s'. Now it's hard to see why one should use this procedure with $S = W$ (as is conventional) rather than with $S = \Phi$. But with $S = \Phi$ rather than $W$, one must factor in the correction term when calculating $p_{S|T,\Gamma}(s \mid t, \gamma)$, and therefore the guessed $\phi$ is different from when $S = W$. If one tries to avoid this change in the guessed $\phi$ by absorbing the correction term into the prior $p_{W|\Gamma}(w \mid \gamma)$, then $p_{\Gamma|T}(\gamma \mid t)$ - which is given by an integral involving that prior - changes. This in turn changes $\gamma$, and therefore the guessed $\phi$ again is different. So presuming one is more directly interested in $\Phi$ rather than $W$, one can't avoid having the correction term affect the evidence procedure.

It should be noted that calculating the correction term can be laborious in large nets. One should bear in mind the determinant-evaluation tricks mentioned in (Buntine and Weigend, 1991), as well as others like the identity $\ln[\; J_{\Phi,W}(w) \;] = Tr(\ln[\; \partial\phi_i / \partial w_j \;]) \cong Tr(\ln^*[\; \partial\phi_i / \partial w_j \;])$, where $\ln^*(.)$ is $\ln(.)$ evaluated to several orders.

## 3      EFFECTS OF THE CORRECTION TERM

To illustrate the effects of the correction term, consider a perceptron with a single output neuron, N input neurons and a unary input space: $o = \tanh(w \cdot x)$, and x always consist of a single one and N - 1 zeroes. For this scenario $\partial\phi_i / \partial w_j$ is an $N \times N$ diagonal matrix, and $\ln[\; J_{\Phi,W}(w) \;] = -2 \Sigma_{k=1}^{N} \ln[\; \cosh(w_k) \;]$. Assume the Gaussian prior and likelihood of the introduction, and for simplicity take $2\sigma^2 = 1$. Both $L(w, t)$ and $L'(w, t)$ are sums of terms each of which only concerns one weight and the corresponding input neuron. Accordingly, it suffices to consider just the i'th weight and the corresponding input neuron.

Let $x(i)$ be the input vector which has its 1 in neuron i. Let $o_j(i)$ be the output of the j'th of the pairs in the training set with input $x(i)$, and $m_i$ the number of such pairs. With $\alpha = 0$

(no weight decay), $L(w, t) = \chi^2(t, w)$, which is minimized by $w'_i = \tanh^{-1}[\sum_{j=1}^{m_i} o_j(i) / m_i]$. If we instead try to minimize $\chi^2(t, w) + J_{\Phi,W}(w)$ though, then for low enough $m_i$ (e.g., $m_i = 1$), we find that there is no minimum. The correction term pushes $w$ away from $0$, and for low enough $m_i$ the likelihood isn't strong enough to counteract this push.

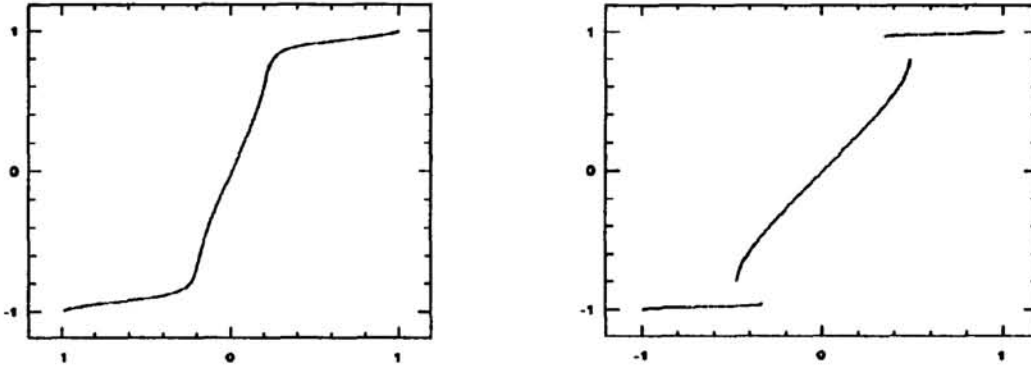

Figures 1 through 3: Train using unmodified BP on training set t, and feed input x into the resultant net. The horizontal axis gives the output you get. If t and x were still used but training had been with modified BP, the output would have been the value on the vertical axis. In succession, the three figures have $\alpha = .6, .4, .4$, and $m = 1, 4, 1$.

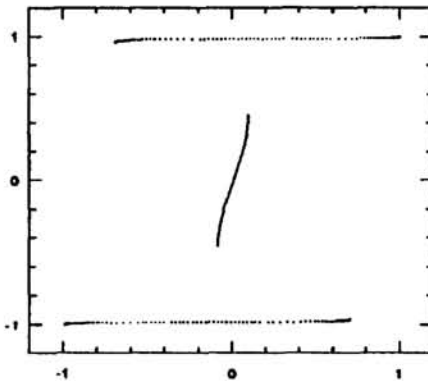

Figure 3.

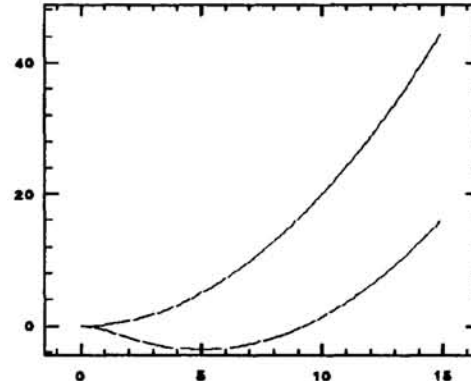

Figure 4: The horizontal axis is $|w_i|$. The top curve depicts the weight decay regularizer, $\alpha w_i^2$, and the bottom curve shows that regularizer modified by the correction term. $\alpha = .2$.

When weight-decay is used though, modified BP finds a solution, just like unmodified BP does. Since the correction term "pushes out" $w$, and since $\tanh(.)$ grows with its argument, a $\phi$ found by modified BP has larger (in magnitude) values of $o$ than does the corresponding $\phi$ found by unmodified BP. In addition, unlike unmodified BP, modified BP has multiple extrema over certain regimes. All of this is illustrated in figures (1) through (3), which graph the value of $o$ resulting from using modified BP with a particular training set t and input value x vs. the value of $o$ resulting from using unmodified BP with t and x. Figure (4) depicts the $w_i$-dependences of the weight decay term and of the weight-decay term plus the correction term. (When there's no data, BP searches for minima of those curves.)

Now consider multi-layer nets, possibly with non-unary X. Denote a vector of the compo-

nents of $w$ which lead from the input layer into hidden neuron K by $w_{[K]}$. Let $x'$ be the input vector consisting of all 0's. Then $\partial \tanh(w_{[K]} \cdot x') / \partial w_j = 0$ for any $j$, $w$, and K, and for any $w$, there is a row of $\partial \phi_i / \partial w_j$ which is all zeroes. This in turn means that $J_{\Phi,W}(w) = 0$ for any $w$, which means $W_{inj}$ is empty, and $p_{\Phi|T}(\phi \mid t)$ is independent of the data $t$. (Intuitively, this problem arises since the $o$ corresponding to $x'$ can't vary with $w$, and therefore the dimension of $\Phi$ is less than |W|) So we must forbid such an all-zeroes $x'$. The easiest way to do this is to require that one input neuron always be on, i.e., introduce a bias unit. An alternative is to redefine $\Phi$ to be the functions from the set $\{X - (0, 0, ..., 0)\}$ to $O$ rather than from the set $X$ to $O$. Another alternative, appropriate when the original $X$ is the set of all input neuron vectors consisting of 0's and 1's, is to instead have input neuron values $\in \{z \ne 0, 1\}$. (In general $z \ne -1$ though; due to the symmetries of the tanh, for many architectures $z = -1$ means that two rows of $\partial \phi_i / \partial w_j$ are identical up to an overall sign, which means that $J_{\Phi,W}(w) = 0$.) This is the solution implicitly assumed from now on.

$J_{\Phi,W}(w)$ will be small - and therefore $p_\Phi(net(w, .))$ will be large - whenever one can make large changes to $w$ without affecting $\phi = net(w, .)$ much. In other words, $p_\Phi(net(w, .))$ will be large whenever we don't need to specify $w$ very accurately. So the correction factor favors those $w$ which can be expressed with few bits. In other words, the correction factor enforces a sort of automatic MDL (Rissanen, 1986; Nowlan and Hinton, 1994).

More generally, for any multi-layer architecture there are many "singular weights" $w_{sin} \notin W_{inj}$ such that $J_{\Phi,W}(w_{sin})$ is not just small but equals zero exactly. $p_W(w)$ must compensate for these singularities, or the peaks of $p_{\Phi|T}(\phi \mid t)$ won't depend on $t$. So we need to have $p_W(w) \to 0$ as $w \to w_{sin}$. Sometimes this happens automatically. For example often $w_{sin}$ includes infinite-valued $w$'s, since $\tanh'(\infty) = 0$. Because $p_W(\infty) = 0$ for the weight-decay prior, that prior compensates for the infinite-$w$ singularities in the correction term.

For other $w_{sin}$ there is no such automatic compensation, and we have to explicitly modify $p_W(w)$ to avoid singularities. In doing so though it seems reasonable to maintain a "bias" towards the $w_{sin}$, that $p_W(w)$ goes to zero slowly enough so that the values $p_\Phi(net(w, .))$ are "enhanced" for $w$ near $w_{sin}$. Although a full characterization of such enhanced $w$ is not in hand, it's easy to see that they include certain kinds of pruned nets (Hassibi and Stork, 1992), weight-shared nets (Nowlan and Hinton, 1994), and feature-selected nets.

To see that (some kinds of) pruned nets have singular weights, let $w^*$ be a weight vector with a zero-valued weight coming out of hidden neuron K. By (1) $p_\Phi(net(w^*, .)) = \int dw' \, p_W(w') \, \delta(net(w', .) - net(w^*, .))$. Since we can vary the value of each weight $w^*_i$ leading into neuron K without affecting $net(w^*, .)$, the integral diverges. So $w^*$ is singular; removing a hidden neuron results in an enhanced probability. This constitutes an *a priori* argument in favor of trying to remove hidden neurons during training.

This argument does not apply to weights leading *into* a hidden neuron; $J_{\Phi,W}(w)$ treats weights in different layers differently. This fact suggests that however $p_W(w)$ compensates for the singularities in $J_{\Phi,W}(w)$, weights in different layers should be treated differently by $p_W(w)$. This is in accord with the advice given in (MacKay, 1992).

To see that some kinds of weight-shared nets have singular weights, let $w'$ be a weight vector such that for any two hidden neurons K and K' the weight from input neuron i to K equals the weight from i to K', for all input neurons i. In other words, $w$ is such that all hid-

den neurons compute identical functions of x. (For some architectures we'll actually only need a single pair of hidden neurons to be identical.) Usually for such a situation there is a pair of columns of the matrix $\partial\phi_i / \partial w_j$ which are exactly proportional to one another. (For example, in a 3-2-1 architecture, with $X = \{z, 1\}^3$, $|W| = |X| \times |O| = 8$, and there are four such pairs of columns.) This means that $J_{\Phi,W}(w') = 0$; w' has an enhanced probability, and we have an *a priori* argument in favor of trying to equate hidden neurons during training.

The argument that feature-selected nets have singular weights is architecture-dependent, and there might be reasonable architectures for which it fails. To illustrate the argument, consider the 3-2-1 architecture. Let $x_1(k)$ and $x_2(k)$ with $k = \{1, 2, 3\}$ designate three distinct pairs of input vectors. For each k have $x_1(k)$ and $x_2(k)$ be identical for all input neurons except neuron A, for which they differ. (Note there are four pairs of input vectors with this property, one for each of the four possible patterns over input neurons B and C.) Let w' be a weight vector such that both weights leaving A equal zero. For this situation net(w', $x_1(k)$) = net(w', $x_2(k)$) for all k. In addition $\partial$ net(w, $x_1(k)$) / $\partial w_j$ = $\partial$ net(w, $x_2(k)$) / $\partial w_j$ for all weights $w_j$ except the two which lead out of A. So k = 1 gives us a pair of rows of the matrix $\partial\phi_i / \partial w_j$ which are identical in all but two entries (one row for $x_1(k)$ and one row for $x_2(k)$). We get another such pair of rows, differing from each other in the exact same two entries, for k = 2, and yet another pair for k = 3. So there is a linear combination of these six rows which is all zeroes. This means that $J_{\Phi,W}(w') = 0$. This constitutes an *a priori* argument in favor of trying to remove input neurons during training.

Since it doesn't favor any $p_W(w)$, the analysis of this paper doesn't favor any $p_\Phi(\phi)$. However when combined with empirical knowledge it *suggests* certain $p_\Phi(\phi)$. For example, there are functions g(w) which empirically are known to be good choices for $p_\Phi$(net(w, .)) (e.g., g(w) $\propto \exp[\alpha w^2]$). There are usually problems with such choices of $p_\Phi(\phi)$ though. For example, these g(w) usually make more sense as a prior over W than as a prior over $\Phi$, which would imply $p_\Phi$(net(w, .)) = g(w) / $J_{\Phi,W}(w)$. Moreover it's empirically true that enhanced w should be favored over other w, as advised by the correction term. So it makes sense to choose a compromise between g(w) and g(w) / $J_{\Phi,W}(w)$. An example is $p_\Phi(\phi) \propto$ g(w) / [ $\lambda_1$ + tanh($\lambda_2 \times J_{\Phi,W}(w)$) ] for two hyperparameters $\lambda_1 > 0$ and $\lambda_2 > 0$.

# 4    BEYOND THE CASE OF BACKPROP WITH |W| = |X| |O|

When O does not approximate a Euclidean vector space, elements of $\Phi$ have probabilities rather than probability densities, and P($\phi$ | t) = $\int$dw $p_{W|T}$(w | t) $\delta$(net(w, .), $\phi$), ($\delta$(., .) being a Kronecker delta function). Moreover, if O is a Euclidean vector space but $|W| > |X| |O|$, then again one must evaluate a difficult integral; $\Phi$ = net(W, .) is not one-to-one so one must use equation (1) rather than (2). Fortunately these two situations are relatively rare.

The final case to consider is $|W| < |X| |O|$ (see section two). Let S(W) be the surface in $\Phi$ which is the image (under net(W, .)) of W. For all $\phi$ $p_\Phi(\phi)$ is either zero (when $\phi \notin$ S(W)) or infinite (when $\phi \in$ S(W)). So as conventionally defined, "MAP $\phi$" is not meaningful.

One way to deal with this case is to embed the net in a larger net, where that larger net's output is relatively insensitive to the values of the newly added weights. An alternative that is applicable when $|W| / |O|$ is an integer is to reduce X by removing "uninteresting" x's. A third alternative is to consider surface densities over S(W), $p_{S(W)}(\phi)$, instead of vol-

ume densities over $\Phi$, $p_\Phi(\phi)$. Such surface densities are given by equation (2), if one uses the metric form of $J_{\Phi,W}(w)$. (Buntine has emphasized that the Jacobian form is not even defined for $|W| < |X| |O|$, since $\partial\phi_i / \partial w_j$ is not square then (personal communication).)

As an aside, note that restricting $p_\Phi(\phi)$ to $S(W)$ is an example of the common theoretical assumption that "target functions" come from a pre-chosen "concept class". In practice such an assumption is usually ludicrous - whenever it is made there is an implicit hope that it constitutes a valid approximation to a more reasonable $p_\Phi(\phi)$.

When decision theory is incorporated into Bayesian analysis, only rarely does it advise us to evaluate an MAP quantity (i.e., use BP). Instead Bayesian decision theory usually advises us to evaluate quantities like $E(\Phi \mid t)$ (Wolpert, 1994). Just as it does for the use of MAP estimators, the analysis of this paper has implications for the use of such $E(\Phi \mid t)$ estimators. In particular, one way to evaluate $E(\Phi \mid t) = \int dw \; p_{W|T}(w \mid t) \; net(w, .)$ is to expand $net(w, .)$ to low order and then approximate $p_{W|T}(w \mid t)$ as a sum of Gaussians (Buntine and Weigend, 1991). Equation (4) suggests that instead we write $E(\Phi \mid t)$ as $\int d\phi \; p_{\Phi|T}(\phi \mid t) \; \phi$ and approximate $p_{\Phi|T}(\phi \mid t)$ as a sum of Gaussians. Since fewer approximations are used (no low order expansion of $net(w, .)$), this might be more accurate.

### Acknowledgements

Thanks to David Rosen and Wray Buntine for stimulating discussion, and to TXN and the SFI for funding. This paper is a condensed version of (Wolpert 1994).

### References

Buntine, W., Weigend, A. (1991). Bayesian back-propagation. *Complex Systems*, 5, p. 603.

Denker, J., LeCun, Y. (1991). Transforming neural-net output levels to probability distributions. In *Neural Information Processing Systems 3*, R. Lippman et al. (Eds).

Fefferman, C. (1993). Reconstructing a neural net from its output. Sarnoff Research Center TR 93-01.

Hassibi, B., and Stork, D. (1992). Second order derivatives for network pruning: optimal brain surgeon. Ricoh Tech Report CRC-TR-9214.

MacKay, D. (1992). Bayesian Interpolation, *and* A Practical Framework for Backpropagation Networks. *Neural Computation*, 4, pp. 415 and 448.

Neal, R. (1993). Bayesian learning via stochastic dynamics. In *Neural Information Processing Systems 5*, S. Hanson et al. (Eds). Morgan Kaufmann.

Nowlan, S., and Hinton, G. (1994). Simplifying Neural Networks by Soft Weight-Sharing. In *Theories of Induction: Proceedings of the SFI/CNLS Workshop on Formal Approaches to Supervised Learning*, D. Wolpert (Ed.). Addison-Wesley, to appear.

Rissanen, J. (1986). Stochastic complexity and modeling. *Ann. Stat.*, 14, p. 1080.

Wolpert, D. (1993). On the use of evidence in neural networks. In *Neural Information Processing Systems 5*, S. Hanson et al. (Eds). Morgan-Kauffman.

Wolpert, D. (1994). Bayesian back-propagation over i-o functions rather than weights. SFI tech. report. ftp'able from archive.cis.ohio-state.edu, as pub/neuroprose/wolpert.nips.93.Z.